# Controlled Recognition Bounds for Visual Learning and Exploration

**Vasiliy Karasev**[1]    **Alessandro Chiuso**[2]    **Stefano Soatto**[1]

[1]University of California, Los Angeles    [2]University of Padova

## Abstract

We describe the tradeoff between the performance in a visual recognition problem and the control authority that the agent can exercise on the sensing process. We focus on the problem of "visual search" of an object in an otherwise known and static scene, propose a measure of control authority, and relate it to the expected risk and its proxy (conditional entropy of the posterior density). We show this analytically, as well as empirically by simulation using the simplest known model that captures the phenomenology of image formation, including scaling and occlusions. We show that a "passive" agent given a training set can provide no guarantees on performance beyond what is afforded by the priors, and that an "omnipotent" agent, capable of infinite control authority, can achieve arbitrarily good performance (asymptotically). In between these limiting cases, the tradeoff can be characterized empirically.

## 1   Introduction

We are interested in visual learning for recognition of objects and scenes embedded in physical space. Rather than using  datasets consisting of collections of isolated snapshots, however, we wish to actively control the sensing process during learning. This is because, in the presence of nuisance factors involving occlusion and scale changes, learning requires mobility [1]. Visual learning is thus a process of discovery, literally *uncovering* occluded portions of an object or scene, and viewing it from close enough that all structural details are revealed.[1] We call this phase of learning *exploration* or *mapping*, accomplished by actively controlling the sensor motion within a scene, or by manipulating an object so as to discover all aspects.[2]

Once exploration has been performed, one has a model (or "map" or "representation") of the scene or object of interest. One can then attempt to detect, localize or recognize a particular object or scene, or a class of them, provided intra-class variability has been exposed during exploration. This phase can yield *localization* – where one wishes to recognize a portion of a mapped scene and, as a byproduct, infer the pose relative to the map – or *search* where a particular object mapped during the exploration phase is detected and localized within an otherwise known scene. This can also be interpreted as a *change detection* problem, where one wishes to revisit a known map to detect changes. In the case

where a known object is sought in an unknown map, exploration and search have to be conducted simultaneously.

Within this scenario, exploration and search can be framed as optimal control and optimal stopping time problems. These relate to active vision (next-best-view generation), active learning, robotic motion planning, sequential decision in the setting of partially-observable Markov decision processes (POMDP) and a number of related fields (including Information Bottleneck, Value of Information) and a vast literature that we cannot extensively review here. As often in this class of problems, inference algorithms are essentially intractable, so we wish to design surrogate tasks and prove performance bounds to ensure desirable properties of the surrogate solution.

In this manuscript we consider the problem of detecting and estimating discrete parameters of an unknown object in a known environment. To this purpose we:

1. Describe the simplest model that includes scaling and occlusion nuisances, a two dimensional "cartoon flatland," and a test suite to perform simulation experiments. We derive an explicit probability model to compute the posterior density given photometric measurements.
2. Discuss the tradeoff between performance in a visual decision task and the *control authority* that the explorer possesses. This tradeoff is akin the tradeoff between rate and distortion in a communication system, but it pertains to decision and control tasks, as opposed to the transmission of data. We characterize this tradeoff for the simple case of a static environment, where control authority relates to reachability and energy.
3. Discuss and test algorithms for visual search based on the maximization of the conditional entropy of future measurements and the proxies of this quantity. These algorithms can be used to locate an unknown object in unknown position of a known environment, or to perform change detection in an otherwise known map, for the purpose of updating it.
4. Provide experimental validation of the algorithms, including *regret* and expected exploration length.

## 1.1 Related prior work

Active search and recognition of objects in the scene has been one of the mainstays of Active Perception in the eighties [2, 3], and has recently resurged (see [4] and references therein). The problem can be formulated as a POMDP [5], solving which requires developing approximate, near-optimal policies. Active recognition using next-best-view generation and object appearance is discussed in [6] where authors use PCA to embed object images in a linear, low dimensional space. The scheme does not incorporate occlusions or scale changes. More recently, information driven sensor control for object recognition was used in [7, 8, 9], who deal with visual and sonar sensors, but take features (*e.g.* SIFT, SURF) to be the observed data. A utility function that accounts for occlusions, viewing angle, and distance to the object is proposed in [10] who aim to actively learn object classifiers during the training stage. Exploration and learning of 3D object surface models by robotic manipulation is discussed in [11]. The case of object localization (and tracking if object is moving) is discussed in [12]; information-theoretic approach for solving this problem using a sensor network is described in [13]. Both authors used realistic, nonlinear sensor models, which however are different from photometric sensors and are not affected by the same nuisances. Typically, information-theoretic utility functions used in these problems are submodular and thus can be efficiently optimized by greedy heuristics [14, 15]. With regards to models, our work is different in several aspects: instead of choosing the next best view on a sphere centered at the object, we model a cluttered environment where the object of interest occupies a negligible volume and is therefore fully occluded when viewed from most locations. Second, we wish to operate in a continuous environment, rather than in a world that is discretized at the outset. Third, given the significance of quantization-scale and occlusions in a visual recognition task, we model the sensing process such that it accounts for both.

## 2 Preliminaries

Let $\mathbf{y} \in \mathcal{Y}$ denote data[3] (measurements) and $\mathbf{x} \in \mathcal{X}$ a hidden class variable from a finite alphabet that we are interested in inferring. If prior $p(\mathbf{x})$ and conditional distributions $p(\mathbf{y}|\mathbf{x})$ are known, the

expected risk can be written as

$$P_e = \int p(y)(1 - \max_i p(x_i|y))dy \qquad (1)$$

and minimized by Bayes' decision rule, which chooses the class label with maximum a posteriori probability. If the distributions above are estimated empirically, the expected risk depends on the data set. We are interested in controlling the data acquisition process so as to make this risk as small as possible. We use the problem of visual search (finding a not previously seen object in a scene) as a motivation. It is related to active learning and experimental design. In order to enforce temporal continuity, we model the search agent ("explorer") as a dynamical system of the form:

$$\begin{cases} \xi_{t+1} = & \xi_t \\ g_{t+1} = & f(g_t, u_t) \\ \mathbf{y}_t = & h(g_t, \xi) + \mathbf{n}_t \end{cases} \qquad (2)$$

where $g_t$ denotes the pose state at time $t$, $u_t$ denotes the control, and $\xi$ denotes the *scene* that describes the search environment – a collection of objects (simply-connected surfaces supporting a radiance function) of which the target $\mathbf{x}$ is one instance. Constraints on the controller enter through $f$; photometric nuisances, quantization and occlusions enter through the measurement map $h$. Additive and unmodeled phenomena that affect observed data are incorporated into $\mathbf{n}_t$, the "noise" term.

## 2.1  Signal models

The simplest model that includes both scaling and occlusion nuisances is the "cartoon flatland", where a bounded subset of $\mathbb{R}^2$ is populated by self-luminous line segments, corresponding to clutter objects. We denote an instance of this model, the scene, by $\xi = (\beta_1, \ldots, \beta_C)$, which is a collection of $C$ objects $\beta_k$. The number of objects in the scene $C$ is the *clutter density* parameter that can possibly grow to be infinite in the limit. Each object is described by its center ($c_k$), length ($l_k$), binary orientation ($o_k$), and radiance function supported on the segment $\rho_k$. This is the "texture" or "appearance" of the object, which in the simplest case can be assumed to be a constant function:

$$\beta_k = (c_k, l_k, o_k, \rho_k) \in [0,1]^3 \times \{0,1\} \times [\mathbb{R}^2 \to \mathbb{R}^+] \qquad (3)$$

An agent can move continuously throughout the search domain. We take the state $g_t \in \mathbb{R}^2$ to be its current position, $u_t \in \mathbb{R}^2$ the currently exerted move, and assume trivial dynamics: $g_{t+1} = g_t + u_t$. More complex agents where $g_t \in SE(3)$ can be incorporated without conceptual difficulties.

The measurement model is that of an omnidirectional $m$-pixel camera, with each entry of $\mathbf{y}_t \in \mathbb{R}^m$ in (2) given by:

$$\mathbf{y}_t(i) = \int_{(i-\frac{1}{2})\frac{2\pi}{m}}^{(i+\frac{1}{2})\frac{2\pi}{m}} \int_0^\infty \rho_{\ell(\theta, g_t)}(z)d\theta d\tau + \mathbf{n}_t(i), \text{ with } z = (\tau\cos(\theta), \tau\sin(\theta)) \qquad (4)$$

where $\frac{2\pi}{m}$ is the angle subtended by each pixel. The integrand is a collection of radiance functions which are supported on objects (line segments). Because of *occlusions*, only the closest objects that intersect the pre-image contribute to the image. The index of the object (clutter or object of interest) that contributes to the image is denoted by $\ell(\theta, g_t)$ and is defined as:

$$\ell(\theta, g_t) = \arg\min_k \left\{ \gamma_k \Big| \exists (s_k, \gamma_k) \in [-\frac{l_k}{2}, \frac{l_k}{2}] \times \mathbb{R}_+ \text{ s.t. } c_k + \begin{pmatrix} o_k \\ 1 - o_k \end{pmatrix} s_k = g + \hat{g}(\theta)\gamma_k \right\} \quad (5)$$

Above, $g$ and $\hat{g}(\theta) = (\cos(\theta), \sin(\theta))$ are current position and direction, respectively. $c_k$, $l_k$, and $o_k$ are $k$-th segment center, length, and orientation. Condition $c_k + s_k = g + \hat{g}(\theta)\gamma_k$ encodes intersection of ray $g + \hat{g}(\theta)$ with a point on a segment $k$. The segment closest to viewer, *i.e.* one that is visible, has the smallest $\gamma_k$. Integration over $\frac{2\pi}{m}$ in (4) accounts for quantization, and the layer model (5) describes occlusions. While the measurement model is non-trivial (in particular, it is not differentiable), it is the simplest that captures the nuisance phenomenology. All unmodeled phenomena are lumped in the additive term $\mathbf{n}_t$, which we assume to be zero-mean Gaussian "noise" with covariance $\sigma^2 I$.

In order to design control sequences to minimize risk, we need to evaluate the uncertainty of future measurements, those we have not yet measured, which are a function of the control action to be taken. To that end, we write the probability model for computing the posterior and the predictive density.

We first describe the general case of *visual exploration* where the environment is unknown. We begin with noninformative prior for objects $k = 1, \ldots, C$

$$p(\boldsymbol{\beta}_k) = p(\mathbf{c}_k)p(\mathbf{l}_k)p(\mathbf{o}_k)p(\boldsymbol{\rho}_k) = U[0, N_c]^2 \times Exp(\lambda) \times Ber(1/2) \times U[0, N_\rho] \qquad (6)$$

where $U, Exp$ and $Ber$ denote uniform, exponential, and Bernoulli distributions parameterized by $N_c$, $\lambda$, and $N_\rho$. Then $p(\boldsymbol{\xi}) = p(\boldsymbol{\beta}_1, ..., \boldsymbol{\beta}_C)$. The posterior is then computed by Bayes rule[4]:

$$p(\boldsymbol{\xi}|y^t, g^t) \propto \prod_{\tau=1}^{t} p(y_\tau|g_\tau, \boldsymbol{\xi})p(\boldsymbol{\xi}) = \prod_{\tau=1}^{t} \mathcal{N}(y_\tau - h(g_\tau, \boldsymbol{\xi}); \sigma^2 I)p(\boldsymbol{\xi}) \qquad (7)$$

Above, $\mathcal{N}(z, \Sigma)$ denotes the value of a zero-mean Gaussian density with covariance $\Sigma$ at $z$. The density can be decomposed as a product of likelihoods since knowledge of environment ($\xi$) and location ($g_t$) is sufficient to predict measurement $y_t$ up to Gaussian noise. The predictive distribution (distribution of the next measurement conditioned on the past) is computed by marginalization:

$$p(\mathbf{y}_{t+1}|y^t, g^t, g_{t+1}) \;=\; \int p(\xi|y^t, g^t, g_{t+1})p(\mathbf{y}_{t+1}|\xi, y^t, g_{t+1})d\xi \qquad (8)$$

$$=\; \int p(\xi|y^t, g^t)\mathcal{N}(\mathbf{y}_{t+1} - h(g_{t+1}, \xi), \sigma^2 I)d\xi \qquad (9)$$

The marginalization above is essentially intractable. In this paper we focus on *visual search* of a particular object in an otherwise known environment, so marginalization is only performed with respect to a single object in the environment, $\mathbf{x}$, whose parameters are discrete, but otherwise analogous to (6):

$$p(\mathbf{x}) = U\{0, ..., N_c - 1\}^2 \times Exp(\lambda) \times Ber(1/2) \times U\{0, \ldots, N_\rho - 1\} \qquad (10)$$

We denote by $x_i$, $i = 1, ..., |\mathcal{X}|$ object with parameters $(c_i, l_i, o_i, \rho_i)$ and write $\xi_i = (x_i, \beta_1, \ldots, \beta_C)$ to denote the scene with *known* clutter objects $\beta_1, ..., \beta_C$ augmented by an *unknown* object $x_i$. In this case, we have:

$$p(\mathbf{x}|y^t, g^t) \propto \prod_{\tau=1}^{t} \mathcal{N}(y_\tau - h(g_\tau, \boldsymbol{\xi}); \sigma^2 I)p(\mathbf{x}) \qquad (11)$$

$$p(\mathbf{y}_{t+1}|y^t, g^t, g_{t+1}) = \sum_{i=1}^{|\mathcal{X}|} p(x_i|y^t, g^t)\mathcal{N}(\mathbf{y}_{t+1} - h(g_{t+1}, \xi_i), \sigma^2 I) \qquad (12)$$

## 3   The role of control in active recognition

It is clear from equations (11) and (12) that the history of agent's positions $g^t$ plays a key role in the process of acquiring new information on the object of interest $\mathbf{x}$ for the purpose of recognition. This is encoded by the conditional density (11). In the context of the identification of the model (2), one would say that data $y^t$ (a function of the scene and the history of positions) must be sufficiently informative [16] on $\mathbf{x}$, meaning that $y^t$ contains enough information to estimate $\mathbf{x}$; this can be measured e.g. through the Fisher information matrix if $\mathbf{x}$ is deterministic but unknown, or by the posterior $p(\mathbf{x}|y^t)$ in a probabilistic setting. This depends upon whether $u^t$ is sufficiently exciting, a "richness" condition that has been extensively used in the identification and adaptive control literature [17, 18], which guarantees that the state trajectory $g^t$ explores the space of interest. If this condition is not satisfied, there are limitations on the performance that can be attained during the search process. There are two extreme cases which set an upper and lower bounds on recognition error:

1. Passive recognition: there is no active control, and instead a collection of vantage points $g^t$ is given a-priori. Under this scenario it is easy to prove that, averaging over the possible scenes and initial agent locations, the probability of error approaches chance (i.e. that given by the prior distribution) as clutter density and/or the environment volume increase.

2. Full control on $g^t$: if the control action can take the "omnipotent agent" anywhere, and infinite time is available to collect measurements, then the conditional entropy $H(\mathbf{x}|y^t)$ decreases asymptotically to zero thus providing arbitrarily good recognition rate in the limit.

In general, there is a tradeoff between the ability to gather new information through suitable control actions, which we name "control authority", and the recognition rate. In the sequel we shall propose a measure for the "control authority" over the sensing process; later in the paper we will consider conditional entropy as a proxy (upper bound) on probability of error and evaluate empirically how control authority affects the conditional entropy decrease.

## 3.1 Control authority

Unlike the passive case, in the controlled scenario time plays an important role. This happens in two ways. One is related to the ability to visit previously unexplored regions and therefore is related to the reachable space under input and time constraints, the other is the effect of noise which needs to be averaged. If objects in the scene move, this can be done only at an expense in energy, and achieving asymptotic performance may not be possible under control limitations. This considerably more complex scenario is beyond our scope in this paper. We focus on the simplest case of static environment.

Control authority depends on (i) the controller $u$, as measured for instance by a norm[5] $\|u\|$ : $\mathcal{U}[0, T] \to \mathbb{R}$ and (ii) on the geometry of the state space, the input-to-state map and on the environment. We propose to measure control authority in the following manner: associate to each pair of locations in the state space $(g_o, g_f)$ and a given time horizon $T$ the cost $\|u\|$ required to move from $g_o$ at time $t = 0$ to $g_f$ at time $t = T$ along a minimum cost path i.e.

$$J_\xi(g_o, g_f, T) \doteq \inf_{u \,:\, g_u(0)=g_o, g_u(T)=g_f \,\xi} \|u\| \tag{13}$$

where $g_u(t)$ is the state vector at time $t$ under control $u$. If $g_f$ is not reachable from $g_o$ in time $T$ we set $J_\xi(g_o, g_f, T) = \infty$. This will depend on the dynamical properties of the agent $\dot{g} = f(g, u)$ (or $g_{t+1} = f(g_t, u_t)$ for discrete time) as well as on the scene $\xi$ where the agent has to navigate through while avoiding obstacles.

The control authority ($\mathcal{CA}$) can be measured via the volume of the reachable space for fixed control cost, and will be a function of the initial configuration $g_0$ and of the scene $\xi$, i.e.

$$\mathcal{CA}(k, g_o, \xi) \doteq Vol\{g_f : J_\xi(g_0, g_f, k) \leq 1\} \tag{14}$$

If instead one is interested in *average* performance (e.g. w.r.t. the possible scene distributions with fixed clutter density), a reasonable measure is the average of smallest volume (as $g_0$ varies) of the reachable space with a unit cost input

$$\mathcal{CA}(k) \doteq E_\xi \Big[\inf_{g_o} \mathcal{CA}(k, g_o, \xi)\Big] \tag{15}$$

If planning on an indefinitely long time horizon is allowed, then one would minimize $J(g_o, g_f, T)$ over time $T$:

$$J(g_o, g_f) \doteq \inf_{T \geq 0} J(g_o, g_f, T) \tag{16}$$

with

$$\mathcal{CA}_\infty \doteq \inf_{g_o} (Vol\{g_f : J(g_o, g_f) \leq 1\}) \tag{17}$$

The figures $\mathcal{CA}(k, g_o, \xi)$ in (14), $\mathcal{CA}(k)$ and $\mathcal{CA}_\infty$ in (17) are proxies of the exploration ability which, in turn, is related to the ability to gather new information on the task at hand. The data acquisition process can be regarded as an experiment design problem [16] where the choice of the control signal guides the experiment. Control authority, as defined above, measures how much freedom one has on the sampling procedure; the larger the $\mathcal{CA}$, the more freedom the designer has. Hence, having fixed (say) the number of snapshots of the scene one may consider the time interval over which these snapshots can be taken, the designer is trying to maximize the information the data contains on the task (making a decision on class label); this information is of course a nondecreasing function of $\mathcal{CA}$. More control authority corresponds to more freedom in the choice of which samples one is taking (from which location and at which scale).

Therefore the risk, considered against $\mathcal{CA}(k)$ in (15), $\mathcal{CA}(k, g_o, \xi)$ in (14) or $\mathcal{CA}_\infty$ in (17) will follow a surface that depends on the clutter: For any given clutter (or clutter density), the risk will be a monotonically non-increasing function of control authority $\mathcal{CA}(k)$. This is illustrated in Fig. 4.

# 4 Control policy

Given $g_t$, $\xi$, and a finite control authority $\mathcal{CA}(k, g_t, \xi)$, in order to minimize average risk (1) with respect to a sequence of control actions we formulate a finite $k$-step horizon optimal control problem:

$$u_t^{*t+k-1} = \arg \min_{u_t^{t+k-1}} \int p(\mathbf{y}_{t+1}^{t+k}|y^t, u_t^{t+k-1})\big(1 - \max_i p(x_i|y^t, \mathbf{y}_{t+1}^{t+k}, u_t^{t+k-1})\big) d\mathbf{y}_{t+1}^{t+k} \qquad (18)$$

which is unfortunately intractable. As is standard, we can settle for the greedy $k = 1$ case:

$$u_t^* = \arg \min_{u_t} \int p(\mathbf{y}_{t+1}|y^t, u_t)\big(1 - \max_i p(x_i|y^t, \mathbf{y}_{t+1}, u_t)\big) d\mathbf{y}_{t+1} \qquad (19)$$

but it is still often impractical. We relax the problem further by choosing to minimize the upper bound on Bayesian risk, of which a convenient one is the conditional entropy (see [19], which shows $P_e \leq \frac{1}{2} H(\mathbf{x}|\mathbf{y})$): This implies that control action can be chosen by entropy minimization:

$$u_t^* = \arg \min_{u_t} H(\mathbf{x}|y^t, \mathbf{y}_{t+1}, u_t) \qquad (20)$$

Using chain rules of entropy, we can rewrite minimization of $H(\mathbf{x}|y^t, \mathbf{y}_{t+1}, u_t)$ as maximization of conditional entropy of next measurement:

$$u_t^* = \arg \min_{u_t} H(\mathbf{x}|y^t, \mathbf{y}_{t+1}, u_t) = \arg \min_{u_t} H(\mathbf{x}|y^t) - I(\mathbf{y}_{t+1}; \mathbf{x}|y^t, u_t) \qquad (21)$$

$$= \arg \max_{u_t} H(\mathbf{y}_{t+1}|y^t, u_t) - H(\mathbf{y}_{t+1}|y^t, u_t, \mathbf{x}) \qquad (22)$$

$$= \arg \max_{u_t} H(\mathbf{y}_{t+1}|y^t, u_t) \qquad (23)$$

because $H(\mathbf{y}_{t+1}|y^t, u_t, \mathbf{x}) = H(\mathbf{n}_t)$ is due to Gaussian noise, since $\mathbf{y}_{t+1} = h(g_{t+1}; \xi) + \mathbf{n}_{t+1}$ and both $g_{t+1}$ and $\xi$ are known (the only unknown object in the scene is $\mathbf{x}$, and it is conditioned on). $H(\mathbf{y}_{t+1}|y^t, u_t)$ is the entropy of a Gaussian mixture distribution which can be easily approximated by Monte Carlo, and for which both lower [20] and upper bounds [21] are known.

Since the controller has energy limitations, *i.e.* is unable to traverse the environment in one step, optimization is taken over a small ball in $\mathbb{R}^2$ centered at current location $g_t$. In practice, the set of controls needs to be discretized and entropy computed for each action. However, rather than myopically choosing the next control, we instead choose the next target position, in a "bandit" approach [22, 23]: maximization in (23) is taken with respect to all locations in the world, rather than the set of controls (locations reachable in one step), and the agent takes the minimum energy path toward the most informative location. Since this location is typically not reachable in a single step, one can adopt a "stubborn" strategy that follows the planned path to the target location before choosing next action, and an "indecisive" – that replans as soon as additional information becomes available as a consequence of motion. We demonstrate the characteristics of conditional entropy as a criterion for planning in Fig. 1.

# 5 Experiments

In addition to evaluating "indecisive" and "stubborn" strategies, we also consider several different uncertainty measures. Section 4 provided arguments for $H(\mathbf{y}_{t+1}|y^t, g)$ (a "max-ent" approach) which is a proxy for minimization of Bayesian risk. Another option is to maximize covariance of $p(\mathbf{y}_{t+1}|y^t, g)$ ("max-var"), for example due to reduced computational cost. Alternatively, if we do not wish to hypothesize future measurements and compute $p(\mathbf{y}_{t+1}|y^t, g)$, we may search by approaching the mode of the posterior distribution $p(\mathbf{x}|y^t)$ ("max-posterior"). To test average performance of these strategies, we consider search in 100 environment instances, each containing 40 known clutter objects and one unknown object. Clutter objects are sampled from the continuous prior distribution (6) and unknown object is chosen from the prior (10) discretized to $|\mathcal{X}| \approx 9000$. Agent's sensor has $m = 30$ pixels, with additive noise $\sigma$ set to half of the difference between object colors. Conditional entropy of the next measurement, $H(\mathbf{y}_{t+1}|y^t, g_{t+1})$, is calculated over the entire map, on a 16x16 grid. Search is terminated once residual entropy falls below a threshold value: $H(\mathbf{x}|y^t) < 0.001$. We are interested in average search time (expressed in terms of number of steps) and average *regret*, which

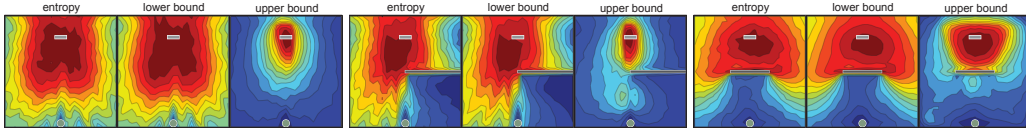

Figure 1: *"Value of measurement" described by conditional entropy $H(\mathbf{y}_{t+1}|y^t, g)$ as a function of location $g$. We focus on three special cases, and for each show entropy, its lower bound (see [20]), and upper bound (based on Gaussian approximation, see [24]). In all cases, the agent is at the bottom of the environment, and a small unknown object is at the top. The agent has made one measurement ($y_1$) and must now determine the best location to visit. The left three panels demonstrate a case of scaling: object is seen, but due to noise and quantization its parameters are uncertain. Agent gains information if putative object location (top) is approached. Middle three panels demonstrate partial occlusion: a part of the object has been seen, and there is now a region (bottom right corner) that is uninformative – measurements taken there are predictable. Full occlusion is shown in the right three panels. The object has not been seen (due to occluder in the middle of the environment) and the best action is to visit new area. Notice that lower and upper bounds are maximized at the same point as actual entropy. This was a common occurrence in many experiments that we did. Because we are interested in the maximizing point, rather than the maximizing value, even if the bounds are loose, using them for navigation can lead to reasonable results.*

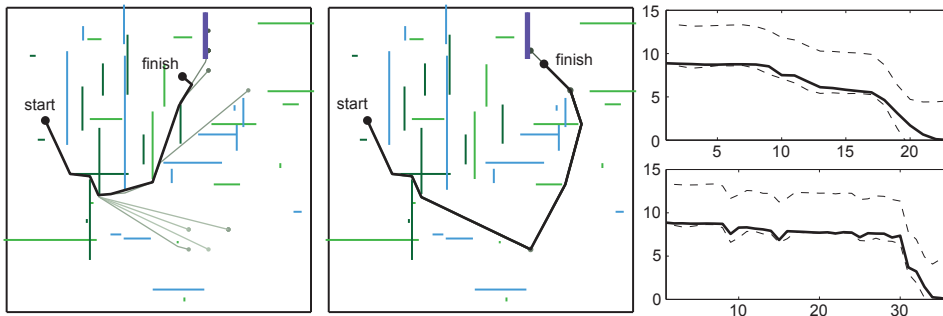

Figure 2: *A typical run of "indecisive" (left) and "stubborn" (right) strategies. Objects are colored according to their radiance and the unknown object is shown as a thick line. Traveled path is shown in black. The thinner lines are the planned paths that were not traversed to the end because of replanning. Stubborn explorer traverses each planned segment to its end. Right: Residual entropy $H(\mathbf{x}|y^t)$ shown over time for the two strategies (top: "indecisive", bottom: "stubborn"). Lower and upper bounds on $H(\mathbf{x}|y^t, \mathbf{y}_{t+1})$ can be computed prior to measuring $\mathbf{y}_{t+1}$ using upper and lower bounds on $H(\mathbf{y}_{t+1}|y^t)$. Sharp decrease occurs when object becomes visible.*

we define as the excess fraction of the minimum energy path to the center of the unknown object ($c_0$) that the explorer takes: $regret \doteq \frac{c_u(x_o, c_0) - J(x_o, c_0)}{J(x_o, c_0)}$. Because it is not always necessary to reach the object to recognize it (viewing it closely from multiple viewpoints may be sufficient), this quantity is an approximation to minimum search effort. We show an example of a typical problem instance in Fig. 2. Statistics of strategies' performance are shown in Fig. 3. Minimum energy path and random walk strategy play roles of lower and upper bounds. For each of the three uncertainty measures, "indecisive" outperformed "stubborn" in terms of both average path length and average regret, as also shown in Table 1. Notice however that for specific problem instances "indecisive" can be much worse than "stubborn" – the curves for the two strategy types cross. Generally, "max-ent" strategy seems to perform best, followed by "max-var", and "max-posterior". "Random-walk" strategy was unable to find the object unless it was visible initially or became visible by chance. We next

| | Average search duration | | | Average regret | | |
|---|---|---|---|---|---|---|
| | max-ent | max-var | max-$p(\mathbf{x}|y^t)$ | max-ent | max-var | max-$p(\mathbf{x}|y^t)$ |
| indecisive | 28.42 | 32.70 | 41.00 | 1.27 | 1.44 | 1.96 |
| stubborn | 34.26 | 36.17 | 41.49 | 1.71 | 1.78 | 2.19 |

Table 1: *Search time statistics for different strategies.*

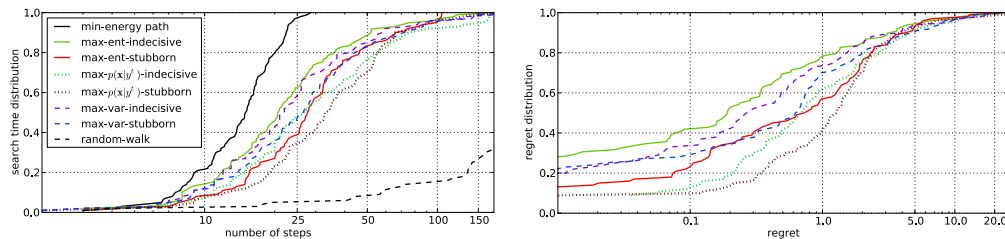

Figure 3: *Search time statistics for a 100 world test suite. Left: cumulative distribution of distance until detection traveled by the max-entropy, max-posterior, max-variance explorers, and random walker. Right: cumulative distribution of regret for the explorers.*

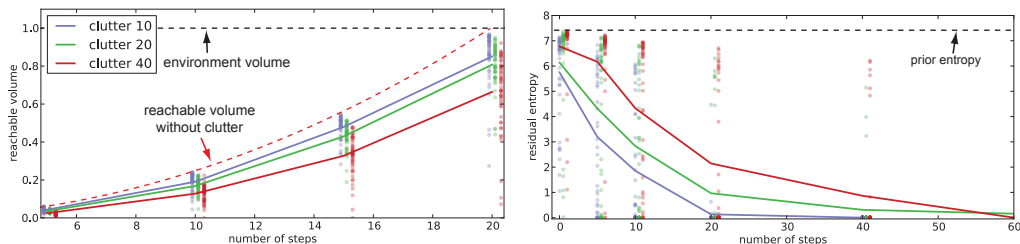

Figure 4: *Left: Control authority. The red dashed curve corresponds to reachable volume in the absence of clutter. The black dashed line is the normalized maximum reachable volume in the environment. Right: Residual entropy $H(\mathbf{x}|y^t)$, as a function of control authority and clutter density. Black dashed line indicates $H(\mathbf{x})$, entropy prior to taking any measurements. Lines correspond to residual entropy for a given control authority averaged over the test suite; markers – to residual entropy on a specific problem instance. For certain scenes, agent is unable to significantly reduce entropy because the object never becomes unoccluded (once object is seen, there is a sharp drop in residual entropy, as shown in Fig. 2).*

empirically evaluated explorer's exploration ability under finite control authority. Reachable volume was computed by Monte Carlo sampling, following (14)-(15) for several clutter density values. For each clutter density, we generated 40 scene instances and tested "indecisive" max-entropy strategy with respect to control authority. Here $|\mathcal{X}| \approx 2000$, and other parameters remained as in previous experiment. Fig. 4 empirically verifies discussion in Section 3.

## 6 Discussion

We have described a simple model that captures the phenomenology of nuisances in a visual search problem, that includes uncertainty due to occlusion, scaling, and other "noise" processes, and used it to compute the entropy of the prediction density to be used as a utility function in the control policy. We have then related the amount of "control authority" the agent can exercise during the data acquisition process with the performance in the visual search task. The extreme cases show that if one is given a passively gathered dataset of an arbitrary number of images, performance cannot be guaranteed beyond what is afforded by the prior. In the limit of infinite control authority, arbitrarily good decision performance can be attained. In between, we have empirically characterized the tradeoff between decision performance and control authority. We believe this to be a natural extension of rate-distortion tradeoffs where the underlying task is not transmission and storage of data, but usage of (visual) data for decision and control.

**Acknowledgments**

Research supported on ARO W911NF-11-1-0391 and DARPA MSEE FA8650-11-1-7154.

## Footnotes

[1]It has been shown [1]  that mobility is required in order to reduce the Actionable Information Gap, the difference between the complexity of a maximal invariant of the data and the minimal sufficient statistic of a complete representation of the underlying scene.

[2]Note that we are not suggesting that one should construct a three-dimensional (3-D) model of an object or a scene for recognition, as opposed to using collections of 2-D images. From an information perspective, there is no gain in replacing a collection of 2-D images with a 3-D model computed from them. What matters is *how these images are collected*. The multiple images must portray *the same scene* or object, lest one cannot attribute the variability in the data to *nuisance factors* as opposed to *intrinsic variability* of the object of interest. The multiple images must enable establishing correspondence between different images of the same scene. Temporal continuity enables that.

[3]Random variables will be displayed in boldface (*e.g.* $\mathbf{y}$), and realizations in regular fonts (*e.g.* $y$).

[4]superscript in *e.g.* $y^t$ indicates history of $y$ up to $t$, *i.e.* $y^t \doteq (y_1, \ldots, y_t)$ and $y_t^{t+T} \doteq (y_t, \ldots y_{t+T})$

[5]This could be, for instance, total energy, (average) power, maximum amplitude and so on. We can assume that the control is such that $\|u\| \leq 1$

# References

[1] S. Soatto. Steps towards a theory of visual information: Active perception, signal-to-symbol conversion and the interplay between sensing and control. *arXiv:1110.2053*, 2011.

[2] R. Bajcsy. Active perception. 76(8):996–1005, 1988.

[3] D. H. Ballard. Animate vision. *Artificial Intelligence*, 48(1):57–86, 1991.

[4] A. Andreopoulos and J. K. Tsotsos. A theory of active object localization. In *Proceedings of the IEEE International Conference on Computer Vision (ICCV)*, 2009.

[5] N. Roy, G. Gordon Gordon, and S. Thrun. Finding approximate POMDP solutions through belief compression. *Journal of Artificial Intelligence Research*, 23:1–40, 2005.

[6] H. Kopp-Borotschnig, L. Paletta, M. Prantl, and A. Pinz. Appearance-based active object recognition. *Image and Vision Computing*, 18(9):715–727, 2000.

[7] R. Eidenberger and J. Scharinger. Active perception and scene modeling by planning with probabilistic 6d object poses. In *Proceedings of the IEEE International Conference on Intelligent Robots and Systems (IROS)*, 2010.

[8] J. Ma and J. W. Burdick. Dynamic sensor planning with stereo for model identification on a mobile platform. In *Proceedings of the IEEE International Conference on Robotics and Automation (ICRA)*, 2010.

[9] G. A. Hollinger, U. Mitra, and G. S. Sukhatme. Active classification: Theory and application to underwater inspection. In *International Symposium on Robotics Research*, 2011.

[10] Z. Jia, A. Saxena, and T. Chen. Robotic object detection: Learning to improve the classifiers using sparse graphs for path planning. In *IJCAI*, 2011.

[11] M. Krainin, B. Curless, and D. Fox. Autonomous generation of complete 3d object models using next best view manipulation planning. In *Proceedings of the IEEE International Conference on Robotics and Automation (ICRA)*, 2011.

[12] F. Bourgault, A. Göktogan, T. Furukawa, and H. F. Durrant-Whyte. Coordinated search for a lost target in a Bayesian world. *Advanced Robotics*, 18(10), 2004.

[13] G. M. Hoffmann and C. J. Tomlin. Mobile sensor network control using mutual information methods and particle filters. *IEEE Transactions on Automatic Control*, 55(1), 2010.

[14] A. Krause and C. Guestrin. Near-optimal nonmyopic value of information in graphical models. In *Uncertainty in Artificial Intelligence*, 2005.

[15] J.L. Williams, J.W. Fisher III, and A.S. Willsky. Performance guarantees for information theoretic active inference. *AI & Statistics (AISTATS)*, 2007.

[16] L. Pronzato. Optimal experimental design and some related control problems. *Automatica*, 44:303–325, 2008.

[17] R. Bitmead. Persistence of excitation conditions and the convergence of adaptive schemes. *Information Theory, IEEE Transactions on*, 30(2):183 – 191, 1984.

[18] L. Ljung. *System Identification, Theory for the User*. Prentice Hall, 1997.

[19] M. E. Hellman and J. Raviv. Probability of error, equivocation and the Chernoff bound. *IEEE Transactions on Information Theory*, 16:368–372, 1970.

[20] J. R. Hershey and P. A. Olsen. Approximating the Kullback Leibler divergence between Gaussian mixture models. *Proceedings of the IEEE International Conference on Acoustics Speech and Signal Processing (ICASSP)*, 4(6), 2007.

[21] M. F. Huber, T. Bailey, Durrant-Whyte H., and U. D. Hanebeck. On entropy approximation for Gaussian mixture random vectors. In *Proceedings of the IEEE International Conference on Multisensor Fusion and Integration for Intelligent Systems (MFI)*, 2008.

[22] L. Valente, R. Tsai, and S. Soatto. Information gathering control via exploratory path planning. In *Proceedings of the Conference on Information Sciences and Systems*. March 2012.

[23] R. Vidal, Omid Shakernia, H. J. Kim, D. H. Shim, and S. Sastry. Probabilistic pursuit-evasion games: theory, implementation, and experimental evaluation. *IEEE Transactions on Robotics*, 18(5), 2002.

[24] T. M. Cover and J. Thomas. *Elements of Information Theory*. Wiley, 1991.

